# Tree-based reparameterization for approximate inference on loopy graphs

**Martin J. Wainwright, Tommi Jaakkola, and Alan S. Willsky**
Department of Electrical Engineering and Computer Science
Massachusetts Institute of Technology
Cambridge, MA 02139
mjwain@mit.edu          tommi@ai.mit.edu          willsky@mit.edu

## Abstract

We develop a tree-based reparameterization framework that provides a new conceptual view of a large class of iterative algorithms for computing approximate marginals in graphs with cycles. It includes *belief propagation* (BP), which can be reformulated as a very local form of reparameterization. More generally, we consider algorithms that perform exact computations over spanning trees of the full graph. On the practical side, we find that such tree reparameterization (TRP) algorithms have convergence properties superior to BP. The reparameterization perspective also provides a number of theoretical insights into approximate inference, including a new characterization of fixed points; and an invariance intrinsic to TRP/BP. These two properties enable us to analyze and bound the error between the TRP/BP approximations and the actual marginals. While our results arise naturally from the TRP perspective, most of them apply in an algorithm-independent manner to any local minimum of the Bethe free energy. Our results also have natural extensions to more structured approximations [e.g., 1, 2].

## 1 Introduction

Given a graphical model, one important problem is the computation of marginal distributions of variables at each node. Although highly efficient algorithms exist for this task on trees, exact solutions are prohibitively complex for more general graphs of any substantial size. This difficulty motivates the use of approximate inference algorithms, of which one of the best-known and most widely studied is *belief propagation* [3], also known as the sum-product algorithm in coding [e.g., 4].

Recent work has yielded some insight into belief propagation (BP). Several researchers [e.g., 5, 6] have analyzed the single loop case, where BP can be reformulated as a matrix powering method. For Gaussian processes on arbitrary graphs, two groups [7, 8] have shown that the means are exact when BP converges. For graphs corresponding to turbo codes, Richardson [9] established the existence of fixed points, and gave conditions for their stability. More recently, Yedidia et al. [1]

showed that BP corresponds to constrained minimization of the Bethe free energy, and proposed extensions based on Kikuchi expansions [10]. Related extensions to BP were proposed in [2]. The paper [1] has inspired other researchers [e.g., 11, 12] to develop more sophisticated algorithms for minimizing the Bethe free energy. These advances notwithstanding, much remains to be understood about the behavior of BP.

The framework of this paper provides a new conceptual view of various algorithms for approximate inference, including BP. The basic idea is to seek a *reparameterization* of the distribution that yields factors which correspond, either exactly or approximately, to the desired marginal distributions. If the graph is acyclic (i.e., a tree), then there exists a unique reparameterization specified by exact marginal distributions over cliques. For a graph with cycles, we consider the idea of iteratively reparameterizing different parts of the distribution, each corresponding to an acyclic subgraph. As we will show, BP can be interpreted in exactly this manner, in which each reparameterization takes place over a pair of neighboring nodes. One of the consequences of this interpretation is a more storage-efficient "message-free" implementation of BP. More significantly, this interpretation leads to more general updates in which reparameterization is performed over arbitrary acyclic subgraphs, which we refer to as *tree-based reparameterization* (TRP) algorithms.

At a low level, the more global TRP updates can be viewed as a tree-based schedule for message-passing. Indeed, a practical contribution of this paper is to demonstrate that TRP updates tend to have better convergence properties than local BP updates. At a more abstract level, the reparameterization perspective provides valuable conceptual insight, including a simple tree-consistency characterization of fixed points, as well as an invariance intrinsic to TRP/BP. These properties allow us to derive an exact expression for the error between the TRP/BP approximations and the actual marginals. Based on this exact expression, we derive computable bounds on the error. Most of these results, though they emerge very naturally in the TRP framework, apply in an algorithm-independent manner to any constrained local minimum of the Bethe free energy, whether obtained by TRP/BP or an alternative method [e.g., 11, 12]. More details of our work can be found in [13, 14].

## 1.1 Basic notation

An undirected graph $\mathcal{G} = (\mathcal{V}, \mathcal{E})$ consists of a set of nodes or vertices $\mathcal{V} = \{1, \ldots, N\}$ that are joined by a set of edges $\mathcal{E}$. Lying at each node $s \in \mathcal{V}$ is a discrete random variable $x_s \in \{0, \ldots, m-1\}$. The underlying sample space $\mathcal{X}^N$ is the set of all $N$ vectors $\mathbf{x} = \{x_s \mid s \in \mathcal{V}\}$ over $m$ symbols, so that $|\mathcal{X}^N| = m^N$. We focus on stochastic processes that are Markov with respect to $\mathcal{G}$, so that the Hammersley-Clifford theorem [ e.g., 3] guarantees that the distribution factorizes as $p(\mathbf{x}) \propto \prod_{\mathcal{C} \in \mathbf{C}} \psi_{\mathcal{C}}(\mathbf{x}_{\mathcal{C}})$ where $\psi_{\mathcal{C}}(\mathbf{x}_{\mathcal{C}})$ is a compatibility function depending only on the subvector $\mathbf{x}_{\mathcal{C}} = \{x_s \mid s \in \mathcal{C}\}$ of nodes in a particular clique $\mathcal{C}$. Note that each individual node forms a singleton clique, so that some of the factors $\psi_{\mathcal{C}}$ may involve functions of each individual variable. As a consequence, if we have independent measurements $y_s$ of $x_s$ at some (or all) of the nodes, then Bayes' rule implies that the effect of including these measurements — i.e., the transformation from the prior distribution $p(\mathbf{x})$ to the conditional distribution $p(\mathbf{x} \mid \mathbf{y})$ — is simply to modify the singleton factors. As a result, throughout this paper, we suppress explicit mention of measurements, since the problem of computing marginals for either $p(\mathbf{x})$ or $p(\mathbf{x} \mid \mathbf{y})$ are of identical structure and complexity. The analysis of this paper is restricted to graphs with singleton ($\psi_s$) and pairwise ($\psi_{st}$) cliques. However, it is straightforward to extend reparameterization to larger cliques, as in cluster variational methods [e.g., 10].

## 1.2 Exact tree inference as reparameterization

Algorithms for optimal inference on trees have appeared in the literature of various fields [e.g., 4, 3]. One important consequence of the junction tree representation [15] is that any exact algorithm for optimal inference on trees actually computes marginal distributions for pairs $(s, t)$ of neighboring nodes. In doing so, it produces an alternative factorization $p(\mathbf{x}) = \prod_{s \in \mathcal{V}} P_s \prod_{(s,t) \in \mathcal{E}} P_{st}/(P_s P_t)$ where $P_s$ and $P_{st}$ are the single-node and pairwise marginals respectively. This $\{P_s, P_{st}\}$ representation can be deduced from a more general factorization result on junction trees [e.g. 15]. Thus, exact inference on trees can be viewed as computing a reparameterized factorization of the distribution $p(\mathbf{x})$ that explicitly exposes the local marginal distributions.

## 2 Tree-based reparameterization for graphs with cycles

The basic idea of a TRP algorithm is to perform successive reparameterization updates on trees embedded within the original graph. Although such updates are applicable to arbitrary acyclic substructures, here we focus on a set $\mathcal{T}^1, \ldots, \mathcal{T}^L$ of embedded spanning trees. To describe TRP updates, let $\mathbf{T}$ be a *pseudo-marginal probability vector* consisting of single-node marginals $T_s(x_s)$ for $s \in \mathcal{V}$; and pairwise joint distributions $T_{st}(x_s, x_t)$ for edges $(s, t) \in \mathcal{E}$. Aside from positivity and normalization ($\sum_{x_s} T_s = 1$; $\sum_{x_s, x_t} T_{st} = 1$) constraints, a given vector $\mathbf{T}$ is arbitrary[1], and gives rises to a parameterization of the distribution as $p(\mathbf{x}; \mathbf{T}) \propto \prod_{s \in \mathcal{V}} T_s \prod_{(s,t) \in \mathcal{E}} T_{st}/\{(\sum_{x_s} T_{st})(\sum_{x_t} T_{st})\}$, where the dependence of $T_s$ and $T_{st}$ on $x$ is omitted for notational simplicity. Ultimately, we shall seek vectors $\mathbf{T}$ that are *consistent* — i.e., that belong to $\mathbb{C} = \{\mathbf{T} \mid \sum_{x_s} T_{st} = T_t \; \forall \; (s, t) \in \mathcal{E}\}$. In the context of TRP, such consistent vectors represent approximations to the exact marginals of the distribution defined by the graph with cycles.

We shall express TRP as a sequence of functional updates $\mathbf{T}^n \mapsto \mathbf{T}^{n+1}$, where superscript $n$ denotes iteration number. We initialize at $\mathbf{T}^0$ via $T_{st}^0 = \kappa \, \psi_s \psi_t \psi_{st}$ and $T_s^0 = \kappa \, \psi_s \prod_{t \in \mathcal{N}(s)} [\sum_{x_t} \psi_{st} \psi_t]$, where $\kappa$ denotes a normalization factor; and $\mathcal{N}(s)$ is the set of neighbors of node $s$. At iteration $n$, we choose some spanning tree $\mathcal{T}^{i(n)}$ with edge set $\mathcal{E}^{i(n)}$, and factor the distribution $p(x; \mathbf{T}^n)$ into a product of two terms

$$p^{i(n)}(\mathbf{x}; \mathbf{T}^n) \quad \propto \quad \prod_{s \in \mathcal{V}} T_s^n \prod_{(s,t) \in \mathcal{E}^{i(n)}} \frac{T_{st}^n}{(\sum_{x_s} T_{st}^n)(\sum_{x_t} T_{st}^n)} \tag{1a}$$

$$r^{i(n)}(\mathbf{x}; \mathbf{T}^n) \quad \propto \quad \prod_{(s,t) \in \mathcal{E}/\mathcal{E}^{i(n)}} \frac{T_{st}^n}{(\sum_{x_s} T_{st}^n)(\sum_{x_t} T_{st}^n)} \tag{1b}$$

corresponding, respectively, to terms in the spanning tree; and residual terms over edges in $\mathcal{E}/\mathcal{E}^{i(n)}$ removed to form $\mathcal{T}^{i(n)}$. We then perform a reparameterization update on $p^{i(n)}(x; \mathbf{T}^n)$ — explicitly:

$$T_{st}^{n+1}(x_s, x_t) \quad = \sum_{\mathbf{x}' \text{ s.t}(x_s', x_t')=(x_s, x_t)} p^{i(n)}(\mathbf{x}'; \mathbf{T}^n) \quad \text{for all } (s, t) \in \mathcal{E}^{i(n)} \tag{2}$$

with a similar update for the single-node marginals $\{T_s \mid s \in \mathcal{V}\}$. These marginal computations can be performed efficiently by any exact tree algorithm applied to $\mathcal{T}^{i(n)}$. Elements of $\mathbf{T}^{n+1}$ corresponding to terms in $r^{i(n)}(x; \mathbf{T}^n)$ are left unchanged

(i.e., $T_{st}^{n+1} = T_{st}^n$ for all $(s,t) \in \mathcal{E}/\mathcal{E}^{i(n)}$). The only restriction placed on the spanning tree set $\mathcal{T}^1, \ldots, \mathcal{T}^L$ is that each edge $(s,t) \in \mathcal{E}$ belong to at least one spanning tree. For practical reasons, it is desirable to choose a set of spanning trees that leads to rapid mixing throughout the graph. A natural choice for the spanning tree index $i(n)$ is the *cyclic ordering*, in which $i(n) \equiv n(\bmod L) + 1$.

### 2.1 BP as local reparameterization

Interestingly, BP can be reformulated in a "message-free" manner as a sequence of local rather than global reparameterization operations. This message-free version of BP directly updates approximate marginals, $T_s$ and $T_{st}$, with initial values determined from the initial messages $M_{st}^0$ and the original compatibility functions of the graphical model as $T_s^0 = \kappa \, \psi_s \prod_{u \in \mathcal{N}(s)} M_{us}^0$ for all $s \in \mathcal{V}$ and $T_{st}^0 = \kappa \, \psi_{st} \psi_s \psi_t \prod_{u \in \mathcal{N}(s)/t} M_{us}^0 \prod_{u \in \mathcal{N}(t)/s} M_{ut}^0$ for all $(s,t) \in \mathcal{E}$, where $\kappa$ denotes a normalization factor. At iteration $n$, these quantities are updated according to the following recursions:

$$T_s^n = \kappa \, T_s^{n-1} \prod_{t \in \mathcal{N}(s)} \frac{1}{T_s^{n-1}} \sum_{x_t} T_{ts}^{n-1} \tag{3a}$$

$$T_{st}^n = \kappa \, \frac{T_{st}^{n-1}}{\left(\sum_{x_s} T_{st}^{n-1}\right)\left(\sum_{x_t} T_{st}^{n-1}\right)} T_s^n T_t^n \tag{3b}$$

The reparameterization form of BP decomposes the graph into a set of two-node trees (one for each edge $(s,t)$); performs exact inference on such tree via equation (3b); and merges the marginals from each tree via equation (3a). It can be shown by induction [see 13] that this simple reparameterization algorithm is equivalent to the message-passing version of BP.

### 2.2 Practical advantages of TRP updates

Since a single TRP update suffices to transmit information globally throughout the graph, it might be expected to have better convergence properties than the purely local BP updates. Indeed, this has proven to be the case in various experiments that we have performed on two graphs (a single loop of 15 nodes, and a $7 \times 7$ grid). We find that TRP tends to converge 2 to 3 times faster than BP on average (rescaled for equivalent computational cost); more importantly, TRP will converge for many problems where BP fails [13]. Further research needs to address the optimal choice of trees (not necessarily spanning) in implementing TRP.

## 3  Theoretical results

The TRP perspective leads to a number of theoretical insights into approximate inference, including a new characterization of fixed points, an invariance property, and error analysis.

### 3.1  Analysis of TRP updates

Our analysis of TRP updates uses a cost function that is an approximation to the Kullback-Leibler divergence between $p(\mathbf{x}; \mathbf{T})$ and $p(\mathbf{x}; \mathbf{U})$ — namely, the quantity

$$G(\mathbf{T};\mathbf{U}) = \sum_{s\in\mathcal{V}} G^s(T_s;U_s) + \sum_{(s,t)\in\mathcal{E}} G^{st}(T_{st};U_{st}) \text{ where}$$

$$G^s(T_s;U_s) = \sum_{x_s} T_s(x_s)\log[T_s(x_s)/U_s(x_s)]$$

$$G^{st}(T_{st};U_{st}) = \sum_{x_s,x_t} T_{st}\Big\{\log[T_{st}/(\sum_{x_s}T_{st})(\sum_{x_t}T_{st})] - \log[U_{st}/(\sum_{x_s}U_{st})(\sum_{x_t}U_{st})]\Big\}$$

Given an arbitrary $\mathbf{U}\in\mathbb{C}$, we show that successive iterates $\{\mathbf{T}^n\}$ of TRP updates satisfy the following *"Pythagorean"* identity:

$$G(\mathbf{U};\mathbf{T}^n) = G(\mathbf{U};\mathbf{T}^{n+1}) + G(\mathbf{T}^{n+1};\mathbf{T}^n) \qquad (4)$$

which can be used to show that TRP fixed points $\mathbf{T}^*$ satisfy the necessary conditions to be local minima of $G$ subject to the constraint $\mathbf{T}^* \in \mathbb{C}$. The cost function $G$, though distinct from the Bethe free energy [1], coincides with it on the constraint set $\mathbb{C}$, thereby allowing us to establish the equivalence of TRP and BP fixed points.

## 3.2 Characterization of fixed points

From the reparameterization perspective arises an intuitive characterization of any TRP/BP fixed point $\mathbf{T}^*$. Shown in Figure 1(a) is a distribution on a graph with

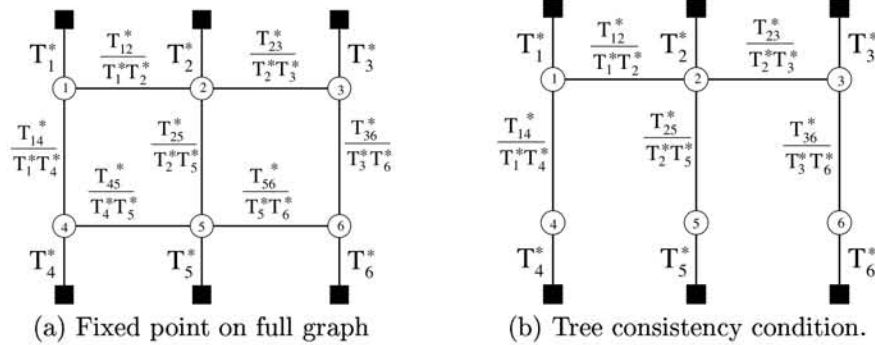

(a) Fixed point on full graph      (b) Tree consistency condition.

**Figure 1.** Illustration of fixed point consistency condition. (a) Fixed point $\mathbf{T}^* = \{T_s^*,\ T_{st}^*\}$ on the full graph with cycles. (b) Illustration of consistency condition on an embedded tree. The quantities $\{T_s^*,\ T_{st}^*\}$ must be *exact* marginal probabilities for any tree embedded within the full graph.

cycles, parameterized according to the fixed point $\mathbf{T}^* = \{T_{st}^*, T_s^*\}$. The consistency condition implies that if edges are removed from the full graph to form a spanning tree, as shown in panel (b), then the quantities $T_{st}^*$ and $T_s^*$ correspond to *exact* marginal distributions over the tree. This statement holds for *any* acyclic substructure embedded within the full graph with cycles — not just the spanning trees $\mathcal{T}^1,\dots,\mathcal{T}^L$ used to implement TRP. Thus, algorithms such as TRP/BP attempt to reparameterize a distribution on a graph with cycles so that it is consistent with respect to each embedded tree.

It is remarkable that the existence of such a parameterization (though obvious for trees) should hold for a positive distribution on an arbitrary graph. Also noteworthy is the parallel to the characterization of max-product[2] fixed points obtained by Freeman and Weiss [16]. Finally, it can be shown [13, 14] that this characterization, though it emerged very naturally from the TRP perspective, applies more generally to any constrained local minimum of the Bethe free energy, whether obtained by TRP/BP, or an alternative technique [e.g., 11, 12].

## 3.3 Invariance of the distribution

A fundamental property of TRP updates is that they leave invariant the full distribution on the graph with cycles. This invariance follows from the decomposition of equation (1): in particular, the distribution $p^{i(n)}(\mathbf{x}; \mathbf{T}^n)$ is left invariant by reparameterization; and TRP does not change terms in $r^{i(n)}(\mathbf{x}; \mathbf{T}^n)$. As a consequence, the overall distribution remains invariant — i.e., $p(\mathbf{x}; \mathbf{T}^n) \equiv p(\mathbf{x}; \mathbf{T}^0)$ for all $n$. By continuity of the map $\mathbf{T} \mapsto p(\mathbf{x}; \mathbf{T})$, it follows that any fixed point $\mathbf{T}^*$ of the algorithm also satisfies $p(\mathbf{x}; \mathbf{T}^*) \equiv p(\mathbf{x}; \mathbf{T}^0)$. This fixed point invariance is also an algorithm-independent result — in particular, all constrained local minima of the Bethe free energy, regardless of how they are obtained, are invariant in this manner [13, 14].

This invariance has a number of important consequences. For example, it places severe restrictions on cases (other than trees) in which TRP/BP can be exact; see [14] for examples. In application to the linear-Gaussian problem, it leads to an elementary proof of a known result [7, 8] — namely, the means must be exact if the BP updates converge.

## 3.4 Error analysis

Lastly, we can analyze the error arising from any TRP/BP fixed point $\mathbf{T}^*$ on an arbitrary graph. Of interest are the exact single-node marginals $P_s$ of the original distribution $p(\mathbf{x}; \mathbf{T}^0)$ defined by the graph with cycles, which by invariance are equivalent to those of $p(\mathbf{x}; \mathbf{T}^*)$. Now the quantities $T_s^*$ have two distinct interpretations: (a) as the TRP/BP approximations to the actual single-node marginals on the full graph; and (b) as the *exact* marginals on any embedded tree (as in Figure 1). This implies that the approximations $T_s^*$ are related to the actual marginals $P_s$ on the full graph by a relatively simple perturbation — namely, removing edges from the full graph to reveal an embedded tree. From this observation, we can derive the following exact expression for the difference between the actual marginal $P_{s;j}$ and the TRP/BP approximation[3] $T_{s;j}^*$:

$$P_{s;j} - T_{s;j}^* = \mathbb{E}_{p^i(\mathbf{x};\mathbf{T}^*)}\left[\left\{\frac{r^i(\mathbf{x};\mathbf{T}^*)}{Z(\mathbf{T}^*)} - 1\right\}\delta(x_s = j)\right] \tag{5}$$

where $i \in \{1, \ldots, L\}$ is an arbitrary spanning tree index; $p^i$ and $r^i$ are defined in equation (1a) and (1b) respectively; $Z(\mathbf{T}^*)$ is the partition function of $p(\mathbf{x}; \mathbf{T}^*)$; $\delta(x_s = j)$ is an indicator function for $x_s$ to take the value $j$; and $\mathbb{E}_{p^i(\mathbf{x};\mathbf{T}^*)}$ denotes expectation using the distribution $p^i(\mathbf{x}; \mathbf{T}^*)$.

Unfortunately, while the tree distribution $p^i(\mathbf{x}; \mathbf{T}^*)$ is tractable, the argument of the expectation includes all terms $r^i(\mathbf{x}; \mathbf{T}^*)$ removed from the original graph to form spanning tree $\mathcal{T}^i$. Moreover, computing the partition function $Z(\mathbf{T}^*)$ is intractable. These difficulties motivate the development of bounds on the error.

In [14], we use convexity arguments to derive a particular set of bounds on the approximation error. Such error bounds, in turn, can be used to compute upper and lower bounds on the actual marginals $P_{s;1}$. Figure 2 illustrates the TRP/BP approximation, as well as these bounds on the actual marginals for a binary process on a $3 \times 3$ grid under two conditions. Note that the tightness of the bounds is closely related to approximation accuracy. Although it is unlikely that these bounds will remain quantitatively useful for general problems on large graphs, they may still yield useful qualitative information.

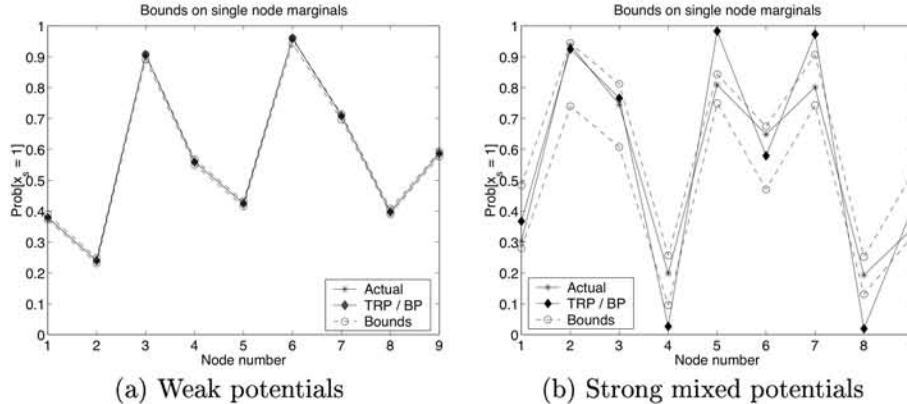

(a) Weak potentials       (b) Strong mixed potentials

**Figure 2.** Behavior of bounds on $3 \times 3$ grid. Plotted are the actual marginals $P_{s;1}$ versus the TRP approximations $T_{s;1}^*$, as well as upper and lower bounds on the actual marginals. (a) For weak potentials, TRP/BP approximation is excellent; bounds on exact marginals are tight. (b) For strong mixed potentials, approximation is poor. Bounds are looser, and for certain nodes, the TRP/BP approximation lies above the upper bounds on the actual marginal $P_{s;1}$.

Much of the analysis of this paper — including reparameterization, invariance, and error analysis — can be extended [see 14] to more structured approximation algorithms [e.g., 1, 2]. Figure 3 illustrates the use of bounds in assessing when to use a more structured approximation. For strong attractive potentials on the $3 \times 3$ grid, the TRP/BP approximation in panel (a) is very poor, as reflected by relatively loose bounds on the actual marginals. In contrast, the Kikuchi approximation in (b) is excellent, as revealed by the tightness of the bounds.

## 4    Discussion

The TRP framework of this paper provides a new view of approximate inference; and makes both practical and conceptual contributions. On the practical side, we find that more global TRP updates tend to have better convergence properties than local BP updates. The freedom in tree choice leads to open problems of a graph-theoretic nature: e.g., how to choose trees so as to guarantee convergence, or to optimize the rate of convergence?

Among the conceptual insights provided by the reparameterization perspective are a new characterization of fixed points; an intrinsic invariance; and analysis of the approximation error. Importantly, most of these results apply to any constrained local minimum of the Bethe free energy, and have natural extensions [see 14] to more structured approximations [e.g., 1, 2].

### Acknowledgments

This work partially funded by ODDR&E MURI Grant DAAD19-00-1-0466; by ONR Grant N00014-00-1-0089; and by AFOSR Grant F49620-00-1-0362; MJW also supported by NSERC 1967 fellowship.

## Footnotes

[1] In general, $\mathbf{T}$ need not be the actual marginals for *any* distribution.

[2]Max-product is a related but different algorithm for computing approximate MAP assignments in graphs with cycles.

[3]The notation $T_{s;j}^*$ denotes the $j^{th}$ element of the vector $T_s^*$.

## References

[1] J. Yedidia, W. T. Freeman, and Y. Weiss. Generalized belief propagation. In *NIPS 13*, pages 689–695. MIT Press, 2001.

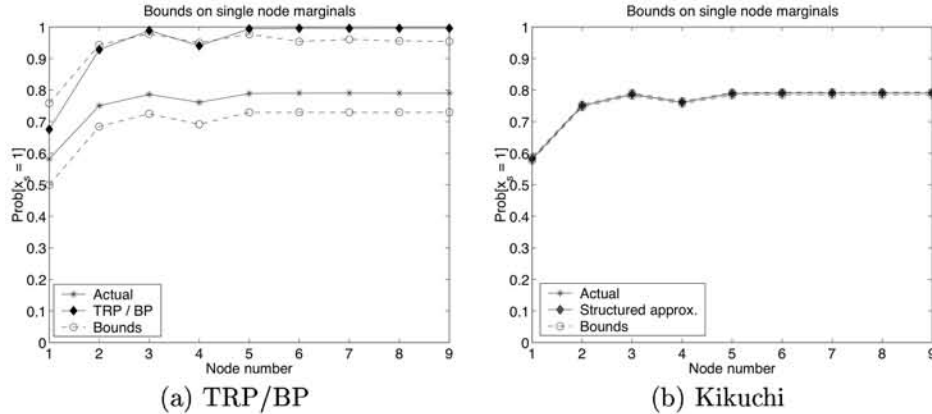

(a) TRP/BP         (b) Kikuchi

**Figure 3.** When to use a more structured approximation? (a) For strong attractive potentials on the $3 \times 3$ grid, BP approximation is poor, as reflected by loose bounds on the actual marginal. (b) Kikuchi approximation [1] for same problem is excellent; corresponding bounds are tight.

[2] T. P. Minka. *A family of algorithms for approximate Bayesian inference*. PhD thesis, MIT Media Lab, 2001.

[3] J. Pearl. *Probabilistic reasoning in intelligent systems*. Morgan Kaufman, San Mateo, 1988.

[4] F. Kschischang and B. Frey. Iterative decoding of compound codes by probability propagation in graphical models. *IEEE Sel. Areas Comm.*, 16(2):219–230, February 1998.

[5] J. B. Anderson and S. M. Hladnik. Tailbiting map decoders. *IEEE Sel. Areas Comm.*, 16:297–302, February 1998.

[6] Y. Weiss. Correctness of local probability propagation in graphical models with loops. *Neural Computation*, 12:1–41, 2000.

[7] Y. Weiss and W. T. Freeman. Correctness of belief propagation in Gaussian graphical models of arbitrary topology. In *NIPS 12*, pages 673–679. MIT Press, 2000.

[8] P. Rusmevichientong and B. Van Roy. An analysis of turbo decoding with Gaussian densities. In *NIPS 12*, pages 575–581. MIT Press, 2000.

[9] T. Richardson. The geometry of turbo-decoding dynamics. *IEEE Trans. Info. Theory*, 46(1):9–23, January 2000.

[10] R. Kikuchi. The theory of cooperative phenomena. *Physical Review*, 81:988–1003, 1951.

[11] M. Welling and Y. Teh. Belief optimization: A stable alternative to loopy belief propagation. In *Uncertainty in Artificial Intelligence*, July 2001.

[12] A. Yuille. A double-loop algorithm to minimize the Bethe and Kikuchi free energies. *Neural Computation*, To appear, 2001.

[13] M. J. Wainwright, T. Jaakkola, and A. S. Willsky. Tree-based reparameterization for approximate estimation on graphs with cycles. LIDS Tech. report P-2510: available at `http://ssg.mit.edu/group/mjwain/mjwain.shtml`, May 2001.

[14] M. Wainwright. *Stochastic processes on graphs with cycles: geometric and variational approaches*. PhD thesis, MIT, Laboratory for Information and Decision Systems, January 2002.

[15] S. L. Lauritzen. *Graphical models*. Oxford University Press, Oxford, 1996.

[16] W. Freeman and Y. Weiss. On the optimality of solutions of the max-product belief propagation algorithm in arbitrary graphs. *IEEE Trans. Info. Theory*, 47:736–744, 2001.
